# Data-Driven Online to Batch Conversions

**Ofer Dekel  and  Yoram Singer**
School of Computer Science and Engineering
The Hebrew University, Jerusalem 91904, Israel
{oferd,singer}@cs.huji.ac.il

## Abstract

Online learning algorithms are typically fast, memory efficient, and simple to implement. However, many common learning problems fit more naturally in the batch learning setting. The power of online learning algorithms can be exploited in batch settings by using *online-to-batch* conversions techniques which build a new batch algorithm from an existing online algorithm. We first give a unified overview of three existing online-to-batch conversion techniques which do not use training data in the conversion process. We then build upon these *data-independent* conversions to derive and analyze *data-driven* conversions. Our conversions find hypotheses with a small risk by explicitly minimizing data-dependent generalization bounds. We experimentally demonstrate the usefulness of our approach and in particular show that the data-driven conversions consistently outperform the data-independent conversions.

## 1  Introduction

*Batch learning* is probably the most common supervised machine-learning setting. In the batch setting, instances are drawn from a domain $\mathcal{X}$ and are associated with target values from a target set $\mathcal{Y}$. The learning algorithm is given a training set of examples, where each example is an instance-target pair, and attempts to identify an underlying rule that can be used to predict the target values of new unseen examples. In other words, we would like the algorithm to *generalize* from the training set to the entire domain of examples. The target space $\mathcal{Y}$ can be either discrete, as in the case of classification, or continuous, as in the case of regression. Concretely, the learning algorithm is confined to a predetermined set of candidate *hypotheses* $\mathcal{H}$, where each hypothesis $h \in H$ is a mapping from $\mathcal{X}$ to $\mathcal{Y}$, and the algorithm must select a "good" hypothesis from $\mathcal{H}$. The quality of different hypotheses in $\mathcal{H}$ is evaluated with respect to a loss function $\ell$, where $\ell(y, y')$ is interpreted as the penalty for predicting the target value $y'$ when the correct target is $y$. Therefore, $\ell(y, h(\mathbf{x}))$ indicates how well hypothesis $h$ performs with respect to the example $(\mathbf{x}, y)$. When $\mathcal{Y}$ is a discrete set, we often use the 0-1 loss, defined by $\ell(y, y') = 1_{y \neq y'}$. We also assume that there exists a probability distribution $\mathcal{D}$ over the product space $\mathcal{X} \times \mathcal{Y}$, and that the training set was sampled i.i.d. from this distribution. Moreover, the existence of $\mathcal{D}$ enables us to reason about the average performance of an hypothesis over its entire domain. Formally, the *risk* of an hypothesis $h$ is defined to be,

$$\mathrm{Risk}_{\mathcal{D}}(h) \;=\; \mathbb{E}_{(\mathbf{x},y) \sim \mathcal{D}}\left[\ell(y, h(\mathbf{x}))\right] \;\;. \tag{1}$$

The goal of a batch learning algorithm is to use the training set to find a hypothesis that does well on average, or more formally, to find $h \in \mathcal{H}$ with a small risk.

In contrast to the batch learning setting, *online learning* takes place in a sequence of rounds. On any given round, $t$, the learning algorithm receives a single instance $\mathbf{x}_t \in \mathcal{X}$ and predicts its target value using an hypothesis $h_{t-1}$, which was generated on the previous round. On the first round, the algorithm uses a default hypothesis $h_0$. Immediately after the prediction is made, the correct target value $y_t$ is revealed and the algorithm suffers an instantaneous loss of $\ell(y_t, h_{t-1}(\mathbf{x}_t))$. Finally, the online algorithm may use the newly obtained example $(\mathbf{x}_t, y_t)$ to improve its prediction strategy, namely to replace $h_{t-1}$ with a new hypothesis $h_t$. Alternatively, the algorithm may choose to stick with its current hypothesis and sets $h_t = h_{t-1}$. An online algorithm is therefore defined by its default hypothesis $h_0$ and the update rule it uses to define new hypotheses. The *cumulative loss* suffered on a sequence of rounds is the sum of instantaneous losses suffered on each one of the rounds in the sequence. In the online setting there is typically no need for any statistical assumptions since there is no notion of generalization. The goal of the online algorithm is simply to suffer a small cumulative loss on the sequence of examples it is given, and examples that are not in this sequence are entirely irrelevant.

Throughout this paper, we assume that we have access to a good online learning algorithm $\mathcal{A}$ for the task on hand. Moreover, $\mathcal{A}$ is computationally efficient and easy to implement. However, the learning problem we face fits much more naturally within the batch learning setting. We would like to develop a batch algorithm $\mathcal{B}$ that exhibits the desirable characteristics of $\mathcal{A}$ but also has good generalization properties. A simple and powerful way to achieve this is to use an *online-to-batch conversion* technique. This is a general name for any technique which uses $\mathcal{A}$ as a building block in the construction of $\mathcal{B}$. Several different online-to-batch conversion techniques have been developed over the years. Littlestone and Warmuth [11] introduced an explicit relation between compression and learnability, which immediately lent itself to a conversion technique for classification algorithms. Gallant [7] presented the *Pocket algorithm*, a conversion of Rosenblatt's online *Perceptron* to the batch setting. Littlestone [10] presented the *Cross-Validation* conversion which was further developed by Cesa-Bianchi, Conconi and Gentile [2]. All of these techniques begin by presenting the training set $(\mathbf{x}_1, y_1), \ldots, (\mathbf{x}_m, y_m)$ to $\mathcal{A}$ in some arbitrary order. As $\mathcal{A}$ performs the $m$ online rounds, it generates a sequence of online hypotheses which it uses to make predictions on each round. This sequence includes the default hypothesis $h_0$ and the $m$ hypotheses $h_1, \ldots, h_m$ generated by the update rule. The aforementioned techniques all share a common property: they all choose $h$, the output of the batch algorithm $\mathcal{B}$, to be one of the online hypotheses $h_0, \ldots, h_m$.

In this paper, we focus on a second family of conversions, which evolved somewhat later and is due to the work of Helmbold and Warmuth [8], Freund and Schapire [6] and Cesa-Bianchi, Conconi and Gentile [2]. The conversion strategies in this family also begin by using $\mathcal{A}$ to generate the sequence of online hypotheses. However, instead of relying on a single hypothesis from the sequence, they set $h$ to be some combination of the entire sequence. Another characteristic shared by these three conversions is that the training data does not play a part in determining how the online hypotheses are combined. That is, the training data is not used in any way other than to generate the sequence $h_0, \ldots, h_m$. In this sense, these conversion techniques are *data-independent*. In this paper, we build on the foundations of these data-independent conversions, and define conversion techniques that explicitly use the training data to derive the batch algorithm from the online algorithm. By doing so, we effectively define the *data-driven* counterparts of the algorithms in [8, 6, 2].

This paper is organized as follows. In Sec. 2 we review the data-independent conversion techniques from [8, 6, 2] and give a simple unified analysis for all three conversions. At the same time, we present a general framework which serves as a building-block for our data-driven conversions. Then, in Sec. 3, we derive three special cases of the general framework

and demonstrate some useful properties of the data-driven conversions. Finally, in Sec. 4, we compare the different conversion techniques on several benchmark datasets and show that our data-driven approach outperforms the existing data-independent approach.

## 2  Voting, Averaging, and Sampling

The first conversion we discuss is the *voting* conversion [6], which applies to problems where the target space $\mathcal{Y}$ is discrete (and relatively small), such as classification problems. The conversion presents the training set $(\mathbf{x}_1, y_1), \ldots, (\mathbf{x}_m, y_m)$ to the online algorithm $\mathcal{A}$, which generates the sequence of online hypotheses, $h_0, \ldots, h_m$. The conversion then outputs the hypothesis $h^{\mathrm{V}}$, which is defined as follows: given an input $\mathbf{x} \in \mathcal{X}$, each online hypothesis casts a vote of $h_i(\mathbf{x})$ and then $h^{\mathrm{V}}$ outputs the target value that receives the highest number of votes. For simplicity, assume that ties are broken arbitrarily. The second conversion is the *averaging* conversion [2] which applies to problems where $\mathcal{Y}$ is a convex set. For example, this conversion is applicable to margin-based online classifiers or to regression problems where, in both cases, $\mathcal{Y} = \mathbb{R}$. This conversion also begins by using $\mathcal{A}$ to generate $h_0, \ldots, h_m$. Then the batch hypothesis $h^{\mathrm{A}}$ is defined to be $\frac{1}{m+1} \sum_{i=0}^{m} h_i(\mathbf{x})$. The third and last conversion discussed here is the *sampling* conversion [8]. This conversion is the most general and applicable to any learning problem, however this generality comes at a price. The resulting hypothesis, $h^{\mathrm{S}}$, is a stochastic function and not a deterministic one. In other words, if applied twice to the same instance, $h^{\mathrm{S}}$ may output different target values. Again, this conversion begins by applying $\mathcal{A}$ to the training set and obtaining the sequence of online hypotheses. Every time $h^{\mathrm{S}}$ is evaluated, it randomly selects one of $h_0, \ldots, h_m$ and uses it to make the prediction. Since $h^{\mathrm{S}}$ is a stochastic function, the definition of $\mathrm{Risk}_{\mathcal{D}}(h^{\mathrm{S}})$ changes slightly and expectation in Eq. (1) is taken also over the random function $h^{\mathrm{S}}$.

Simple data-dependent bounds on the risk of $h^{\mathrm{V}}$, $h^{\mathrm{A}}$ and $h^{\mathrm{S}}$ can be derived, and these bounds are special cases of the more general analysis given below. We now describe a simple generalization of these three conversion techniques. It is reasonable to assume that some of the online hypotheses generated by $\mathcal{A}$ are better than others. For instance, the default hypothesis $h_0$ is determined without observing even a single training example. This surfaces the question whether it is possible to isolate the "best" online hypotheses and only use them to define the batch hypothesis. Formally, let $[m]$ denote the set $\{0, \ldots, m\}$ and let $I$ be some non-empty subset of $[m]$. Now define $h_I^{\mathrm{V}}(\mathbf{x})$ to be the hypothesis which performs voting as described above, with the single difference that only the members of $\{h_i : i \in I\}$ participate in the vote. Similarly, define $h_I^{\mathrm{A}}(\mathbf{x}) = (1/|I|) \sum_{i \in I} h_i(\mathbf{x})$, and let $h_I^{\mathrm{S}}$ be the stochastic function that randomly chooses a function from the set $\{h_i : i \in I\}$ every time it is evaluated, and predicts according to it. The data-independent conversions presented in the beginning of this section are obtained by setting $I = [m]$. Our idea is to use the training data to find a set $I$ which induces the batch hypotheses $h_I^{\mathrm{V}}$, $h_I^{\mathrm{A}}$, and $h_I^{\mathrm{S}}$ with the smallest risk.

Since there is an exponential number of potential subsets of $[m]$, we need to restrict ourselves to a smaller set of candidate sets. Formally, let $\mathcal{I}$ be a family of subsets of $[m]$, and we restrict our search for $I$ to the family $\mathcal{I}$. Following in the footsteps of [2], we make the simplifying assumption that none of the sets in $\mathcal{I}$ include the largest index $m$. This is a technical assumption which can be relaxed at the price of a slightly less elegant analysis. We use two intuitive concepts to guide our search for $I$. First, for any set $J \subseteq [m-1]$, define $L(J) = (1/|J|) \sum_{j \in J} \ell(y_{j+1}, h_j(\mathbf{x}_{j+1}))$. $L(J)$ is the empirical evaluation of the loss of the hypotheses indexed by $J$. We would like to find a set $J$ for which $L(J)$ is small since we expect that good empirical loss of the online hypotheses indicates a low risk of the batch hypothesis. Second, we would like $|J|$ to be large so that the presence of a few bad online hypotheses in $J$ will not have a devastating effect on the performance of the batch hypothesis. The trade-off between these two competing concepts can be formalized

as follows. Let $C$ be a non-negative constant and define,

$$\beta(J) \;=\; L(J) + C\,|J|^{-\frac{1}{2}} \;.\tag{2}$$

The function $\beta$ decreases as the average empirical loss $L(J)$ decreases, and also as $|J|$ increases. It therefore captures the intuition described above. The function $\beta$ serves as our yardstick when evaluating the candidates in $\mathcal{I}$. Specifically, we set $I \;=\; \arg\min_{J \in \mathcal{I}} \beta(J)$. Below we formally justify our choice of $\beta$, and specifically show that $\beta(J)$ is a rather tight upper bound on the risk of $h_J^{\mathrm{A}}$, $h_J^{\mathrm{V}}$ and $h_J^{\mathrm{S}}$. The first lemma relates the risk of these functions with the average risk of the hypotheses indexed by $J$.

**Lemma 1.** *Let* $(\mathbf{x}_1, y_1), \ldots, (\mathbf{x}_m, y_m)$ *be a sequence of examples which is presented to the online algorithm* $\mathcal{A}$ *and let* $h_0, \ldots, h_m$ *be the resulting sequence of online hypotheses. Let* $J$ *be a non-empty subset of* $[m-1]$ *and let* $\ell : \mathcal{Y} \times \mathcal{Y} \to \mathbb{R}_+$ *be a loss function.* **(1)** *If* $\ell$ *is the 0-1 loss then* $\mathrm{Risk}_{\mathcal{D}}(h_J^{\mathrm{V}}) \leq (2/|J|) \sum_{i \in J} \mathrm{Risk}_{\mathcal{D}}(h_i(\mathbf{x}))$. **(2)** *If* $\ell$ *is convex in its second argument then* $\mathrm{Risk}_{\mathcal{D}}(h_J^{\mathrm{A}}) \leq (1/|J|) \sum_{i \in J} \mathrm{Risk}_{\mathcal{D}}(h_i(\mathbf{x}))$. **(3)** *For any loss function* $\ell$ *it holds that* $\mathrm{Risk}_{\mathcal{D}}(h_J^{\mathrm{S}}) = (1/|J|) \sum_{i \in J} \mathrm{Risk}_{\mathcal{D}}(h_i(\mathbf{x}))$.

*Proof.* Beginning with the voting conversion, recall that the loss function being used is the 0-1 loss, namely there is a single correct prediction which incurs a loss of 0 and every other prediction incurs a loss of 1. For any example $(\mathbf{x}, y)$, if more than half of the hypotheses in $\{h_i\}_{i \in J}$ predict the correct outcome then clearly $h_J^{\mathrm{V}}$ also predicts this outcome and $\ell(y, h_J^{\mathrm{V}}(\mathbf{x})) = 0$. Therefore, if $\ell(y, h_J^{\mathrm{V}}(\mathbf{x})) = 1$ then at least half of the hypotheses in $\{h_i\}_{i \in J}$ make incorrect predictions and $(|J|/2) \leq \sum_{i \in J} \ell(y, h_i(\mathbf{x}))$. We therefore get,

$$\ell(y, h_J^{\mathrm{V}}(\mathbf{x})) \;\leq\; \frac{2}{|J|} \sum_{i \in J} \ell(y, h_i(\mathbf{x})) \;.$$

The above holds for any example $(\mathbf{x}, y)$ and therefore also holds after taking expectations on both sides of the inequality. The bound now follows from the linearity of expectation and the definition of the risk function in Eq. (1).

Moving on to the second claim of the lemma, we assume that $\ell$ is convex in its second argument. The claim now follows from a direct application of Jensen's inequality.

Finally, $h_J^{\mathrm{S}}$ chooses its outcome by randomly choosing an hypothesis in $\{h_i : i \in J\}$, where the probability of choosing each hypothesis in this set equals $(1/|J|)$. Therefore, the expected loss suffered by $h_J^{\mathrm{S}}$ on an example $(\mathbf{x}, y)$ is $(1/|J|) \sum_{i \in J} \ell(y, h_i(\mathbf{x}))$. The risk of $h_J^{\mathrm{S}}$ is simply the expected value of this term with respect to the random selection of $(\mathbf{x}, y)$. Again using the linearity of expectation, we obtain the third claim of the lemma. □

The next lemma relates the average risk of the hypotheses indexed by $J$ with the empirical performance of these hypotheses, $L(J)$. In the following lemma, we use capital letters to emphasize that we are dealing with random variables.

**Lemma 2.** *Let* $(X_1, Y_1), \ldots, (X_m, Y_m)$ *be a sequence of examples independently sampled according to* $\mathcal{D}$. *Let,* $H_0, \ldots, H_m$ *be the sequence of online hypotheses generated by* $\mathcal{A}$ *while observing this sequence of examples. Assume that the loss function* $\ell$ *is upper-bounded by* $R$. *Then for any* $J \subseteq [m-1]$,

$$\Pr\left[ \frac{1}{|J|} \sum_{i \in J} \mathrm{Risk}_{\mathcal{D}}(H_i) \;>\; \beta(J) \right] \;<\; \exp\left( -\frac{C^2}{2R^2} \right) \;,$$

*where* $C$ *is the constant used in the definition of* $\beta$ *(Eq. (2)).*

The proof of this lemma is a direct application of Azuma's bound on the concentration of Lipschitz martingales [1], and is identical to that of Proposition 1 in [2]. For concreteness,

we now focus on the averaging conversion and note that the analyses of the other two conversion strategies are virtually identical. By combining the first claim of Lemma 1 with Lemma 2, we get that for any $J \in \mathcal{I}$ it holds that $\text{Risk}_{\mathcal{D}}(h_J^{\text{A}}) \leq \beta(J)$ with probability at least $1 - \exp\left(-C^2/(2R^2)\right)$. Using the union bound, $\text{Risk}_{\mathcal{D}}(h_J^{\text{A}}) \leq \beta(J)$ for all $J \in \mathcal{I}$ simultaneously with probability at least,

$$ 1 - |\mathcal{I}| \exp\left(-\frac{C^2}{2R^2}\right) \ . $$

The greater the value of $C$, the more $\beta$ is influenced by the term $|J|$. On the other hand, a large value of $C$ increases the probability that $\beta$ indeed upper bounds $\text{Risk}_{\mathcal{D}}(h_J^{\text{A}})$ for all $J \in \mathcal{I}$. In conclusion, we have theoretically justified our choice of $\beta$ in Eq. (2).

## 3 Concrete Data-Driven Conversions

In this section we build on the ideas of the previous section and derive three concrete data-driven conversion techniques.

**Suffix Conversion:** An intuitive argument against selecting $I = [m]$, as done by the data-independent conversions, is that many online algorithms tend to generate bad hypotheses during the first few rounds of learning. As previously noted, the default hypothesis $h_0$ is determined without observing any training data, and we should expect the first few online hypotheses to be inferior to those that are generated further along. This argument motivates us to consider subsets $J$ of the form $\{a, a+1, \ldots, m-1\}$, where $a$ is a positive integer less than or equal to $m-1$. Li [9] proposed this idea in the context of the voting conversion and gave a heuristic criterion for choosing $a$. Our formal setting gives a different criterion for choosing $a$. In this conversion we define $\mathcal{I}$ to be the set of all suffixes of $[m-1]$. After the algorithm generates $h_0, \ldots, h_m$, we set $I$ to be $I = \arg\min_{J \in \mathcal{I}} \beta(J)$.

**Interval Conversion:** Kernel-based hypotheses are functions that take the form, $h(\mathbf{x}) = \sum_{j=1}^{n} \alpha_j K(\mathbf{z}_j, \mathbf{x})$, where $K$ is a Mercer kernel, $\mathbf{z}_1, \ldots, \mathbf{z}_n$ are instances, often referred to as *support patterns* and $\alpha_1, \ldots, \alpha_n$ are real weights. A variety of different batch algorithms produce kernel-based hypotheses, including the Support Vector Machine [12]. An important learning problem, which is currently addressed by only a handful of algorithms, is to learn a kernel-based hypothesis $h$ which is defined by at most $B$ support patterns. The parameter $B$ is a predefined constant often referred to as the *budget* of support patterns. Naturally, kernel-based hypotheses which are represented by a few support patterns are memory efficient and faster to calculate. A similar problem arises in the online learning setting where the goal is to construct online algorithms where each online hypothesis $h_i$ is a kernel-based function defined by at most $B$ vectors. Several online algorithms have been proposed for this problem [4, 13, 5]. First note that the data-independent conversions, with $I = [m]$, are inadequate for this setting. Although each individual online hypothesis is defined by at most $B$ vectors, $h^{\text{A}}$ is defined by the union of these sets, which can be much larger than $B$.

To convert a budget-constrained online algorithm $\mathcal{A}$ into a budget-constrained batch algorithm, we make an additional assumption on the update strategy employed by $\mathcal{A}$. We assume that whenever $\mathcal{A}$ updates its online hypothesis, it adds a single new support pattern into the set used to represent the kernel hypothesis, and possibly removes some other pattern from this set. The algorithms in [4, 13, 5] all fall into this category. Therefore, if we choose $I$ to be the set $\{a, a+1, \ldots, b\}$ for some integers $0 \leq a < b < m$, and $\mathcal{A}$ updates its hypothesis $k$ times during rounds $a+1$ through $b$, then $h_I^{\text{A}}$ is defined by at most $B+k$ support patterns. Concretely, define $\mathcal{I}$ to be the set of all non-empty intervals in $[m-1]$. With $C$ set properly, $\beta(J)$ bounds $\text{Risk}_{\mathcal{D}}(h_J^{\text{A}})$ for every $J \in \mathcal{I}$ with high probability. Next,

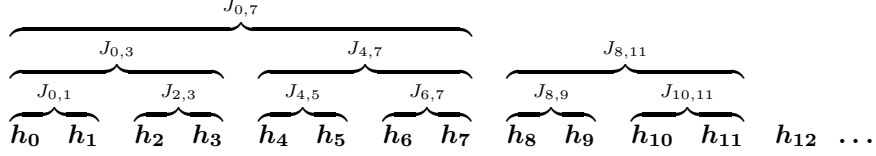

Figure 1: An illustration of the tree-based conversion.

generate $h_0, \ldots, h_m$ by running $\mathcal{A}$ with a budget parameter of $B/2$. Finally, choose $I$ to be the set in $\mathcal{I}$ which contains at most $B/2$ updates and also minimizes the $\beta$ function. By construction, the resulting hypothesis, $h_I^{\mathrm{A}}$, is defined using at most $B$ support patterns.

**Tree-Based Conversion:** A drawback of the suffix conversions is that it must be performed in two consecutive stages. First $h_0, \ldots, h_m$ are generated and stored in memory. Only then can we calculate $\beta(J)$ for every $J \in \mathcal{I}$ and perform the conversion. Therefore, the memory requirements of this conversions grow linearly with $m$. We now present a conversion that can sidestep this problem by interleaving the conversion with the online hypothesis generation. This conversion slightly deviates from the general framework described in the previous section: instead of predefining a set of candidates $\mathcal{I}$, we construct the optimal subset $I$ in a recursive manner. As a consequence, the analysis in the previous section does not directly provide a generalization bound for this conversion. Assume for a moment that $m$ is a power of 2. For all $0 \leq a \leq m-1$ define $J_{a,a} = \{a\}$. Now, assume that we have already constructed the sets $J_{a,b}$ and $J_{c,d}$, where $a, b, c, d$ are integers such that $a < d$, $b = (a + d - 1)/2$, and $c = b + 1$. Given these sets, define $J_{a,d}$ as follows:

$$J_{a,d} = \begin{cases} J_{a,b} & \text{if } \beta(J_{a,b}) \leq \beta(J_{c,d}) \ \wedge \ \beta(J_{a,b}) \leq \beta(J_{a,b} \cup J_{c,d}) \\ J_{c,d} & \text{if } \beta(J_{c,d}) \leq \beta(J_{a,b}) \ \wedge \ \beta(J_{c,d}) \leq \beta(J_{a,b} \cup J_{c,d}) \\ J_{a,b} \cup J_{c,d} & \text{otherwise} \end{cases} \quad . \quad (3)$$

Finally, define $I = J_{0,m-1}$ and output the batch hypothesis $h_I^{\mathrm{A}}$. An illustration of this process is given in Fig. 1. Note that the definition of $I$ requires only $m-1$ recursive evaluations of Eq. (3). When $m$ is not a power of 2, we can pad the sequence of online hypotheses with virtual hypotheses, each of which attains an infinite loss. This conversion can be performed in parallel with the online rounds since on round $t$ we already have all of the information required to calculate $J_{a,b}$ for all $b < t$.

In the special case where the instances are vectors in $\mathbb{R}^n$, $h_0, \ldots, h_m$ are linear hypotheses and we use the averaging technique, the implementation of the tree-based conversion becomes memory efficient. Specifically, assume that each $h_i$ takes the form $h_i(\mathbf{x}) = \mathbf{w}_i \cdot \mathbf{x}$ where $\mathbf{w}_i$ is a vector of weights in $\mathbb{R}^n$. In this case, storing an online hypothesis $h_i$ is equivalent to storing its weight vector $\mathbf{w}_i$. For any $J \subseteq [m-1]$, storing $\sum_{j \in J} h_j$ requires storing the single $n$-dimensional vector $\sum_{j \in J} \mathbf{w}_j$. Hence, once we calculate $J_{a,b}$ we can discard the original online hypotheses $h_a, \ldots, h_b$ and instead merely keep $h_{J_{a,b}}^{\mathrm{A}}$. Moreover, in order to calculate $\beta$ we do not need to keep the set $J_{a,b}$ itself but rather the values $L(J_{a,b})$ and $|J_{a,b}|$. Overall, storing $h_{J_{a,b}}^{\mathrm{A}}$, $L(J_{a,b})$, and $|J_{a,b}|$ requires only a constant amount of memory. It can be verified using an inductive argument that the overall memory utilization of this conversion is $O(\log(m))$, which is significantly less than the $O(m)$ space required by the suffix conversion.

## 4 Experiments

We now turn to an empirical evaluation of the averaging and voting conversions. We chose multiclass classification as the underlying task and used the multiclass version of

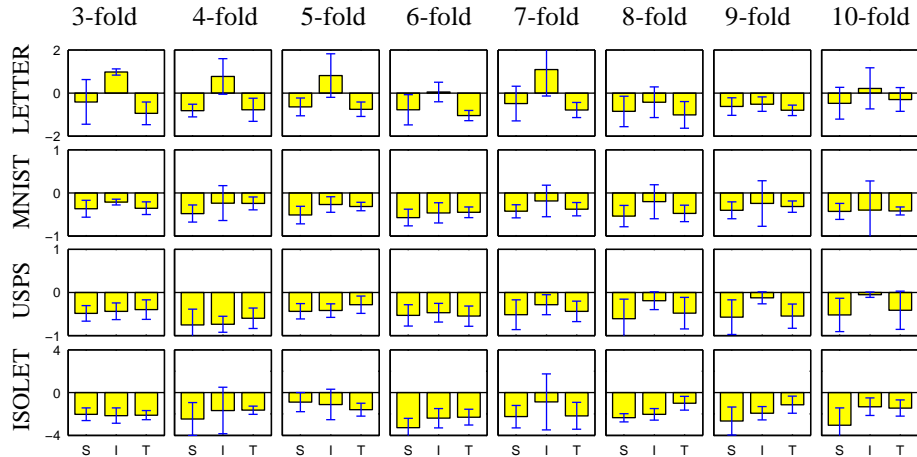

Figure 2: Comparison of the three data-driven averaging conversions with the data-independent averaging conversion, for different datasets (Y-axis) and different training-set sizes (X-axis). Each bar shows the difference between the error percentages of a data-driven conversion (*suffix* (S), *interval* (I) or *tree-based* (T)) and of the data-independent conversion. Error bars show standard deviation over the $k$ folds.

the *Passive-Aggressive* (PA) algorithm [3] as the online algorithm. The PA algorithm is a kernel-based large-margin online classifier. To apply the voting conversion, $\mathcal{Y}$ should be a finite set. Indeed, in multiclass categorization problems the set $\mathcal{Y}$ consists of all possible labels. To apply the averaging conversion $\mathcal{Y}$ must be a convex set. To achieve this, we use the fact that PA associates a margin value with each class, and define $\mathcal{Y} = \mathbb{R}^s$ (where $s$ is the number of classes).

In our experiments, we used the datasets LETTER, MNIST, USPS (training set only), and ISOLET. These datasets are of size 20000, 70000, 7291 and 7797 respectively. MNIST and USPS both contain images of handwritten digits and thus induce 10-class problems. The other datasets contain images (LETTER) and utterances (ISOLET) of the English alphabet. We did not use the standard splits into training set and test set and instead performed cross-validation in all of our experiments. For various values of $k$, we split each dataset into $k$ parts, trained each algorithm using each of these parts and tested on the $k - 1$ remaining parts. Specifically, we ran this experiment for $k = 3, \ldots, 10$. The reason for doing this is that the experiment is most interesting when the training sets are small and the learning task becomes difficult.

We applied the data-independent averaging and voting conversions, as well as the three data-driven variants of these conversions (6 data-driven conversions in all). The interval conversion was set to choose an interval containing 500 updates. The parameter $C$ was arbitrarily set to 3. Additionally, we evaluated the test error of the last hypothesis generated by the online algorithm, $h_m$. It is common malpractice amongst practitioners to use $h_m$ as if it were a batch hypothesis, instead of using an online-to-batch conversion. As a byproduct of our experiments, we show that $h_m$ performs significantly worse than any of the conversion techniques discussed in this paper. The kernel used in all of the experiments is the Gaussian kernel with default kernel parameters. We would like to emphasize that our goal was not to achieve state-of-the-art results on these datasets but rather to compare the different conversion strategies on the same sequence of hypotheses. To achieve the best results, one would have to tune $C$ and the various kernel parameters.

The results for the different variants of the averaging conversion are depicted in Fig. 2.

|  | last | average | average-sfx | voting | voting-sfx |
|---|---|---|---|---|---|
| LETTER 5-fold | $29.9 \pm 1.8$ | $21.2 \pm 0.5$ | $\mathbf{20.5} \pm 0.6$ | $23.4 \pm 0.8$ | $21.5 \pm 0.8$ |
| LETTER 10-fold | $37.3 \pm 2.1$ | $26.9 \pm 0.7$ | $\mathbf{26.5} \pm 0.6$ | $30.2 \pm 1.0$ | $27.9 \pm 0.6$ |
| MNIST 5-fold | $7.2 \pm 0.5$ | $5.9 \pm 0.4$ | $\mathbf{5.3} \pm 0.6$ | $7.0 \pm 0.5$ | $6.5 \pm 0.5$ |
| MNIST 10-fold | $13.8 \pm 2.3$ | $9.5 \pm 0.8$ | $9.1 \pm 0.8$ | $8.7 \pm 0.5$ | $\mathbf{8.0} \pm 0.5$ |
| USPS 5-fold | $9.7 \pm 1.0$ | $7.5 \pm 0.4$ | $\mathbf{7.1} \pm 0.4$ | $9.4 \pm 0.4$ | $8.8 \pm 0.3$ |
| USPS 10-fold | $12.7 \pm 4.7$ | $10.1 \pm 0.7$ | $\mathbf{9.5} \pm 0.8$ | $12.5 \pm 1.0$ | $11.3 \pm 0.6$ |
| ISOLET 5-fold | $20.1 \pm 3.8$ | $17.6 \pm 4.1$ | $\mathbf{16.7} \pm 3.3$ | $20.6 \pm 3.4$ | $18.3 \pm 3.9$ |
| ISOLET 10-fold | $28.6 \pm 3.6$ | $25.8 \pm 2.8$ | $\mathbf{22.7} \pm 3.3$ | $29.3 \pm 3.1$ | $26.7 \pm 4.0$ |

Table 1: Percent of errors averaged over the $k$ folds with standard deviation. Results are given for the last online hypothesis ($h_m$), the data-independent averaging and voting conversions, and their suffix variants. The lowest error on each row is shown in bold.

For each dataset and each training-set size, we present a bar-plot which represents by how much each of the data-driven averaging conversions improves over the data-independent averaging conversion. For instance, the left bar in each plot shows the difference between the test errors of the suffix conversion and the data-independent conversion. A negative value means that the data-driven technique outperforms the data-independent one. The results clearly indicate that the suffix and tree-based conversions consistently improve over the data-independent conversion. The interval conversion does not improve as much and occasionally even looses to the data-independent conversion. However, this is a small price to pay in situations where it is important to generate a compact kernel-based hypothesis. Due to the lack of space, we omit a similar figure for the voting conversion and merely note that the plots are very similar to the ones in Fig. 2.

In Table 1 we give some concrete values of test error, and compare data-independent and data-driven versions of averaging and voting, using the suffix conversion. As a reference, we also give the results obtained by the last hypothesis generated by the online algorithm. In all of the experiments, the data-driven conversion outperforms the data-independent conversion. In general, averaging exhibits better results than voting, while the last online hypothesis is almost always inferior to all of the online-to-batch conversions.

## References

[1] K. Azuma. Weighted sums of certain dependent random variables. *Tohoku Mathematical Journal*, 68:357–367, 1967.

[2] N. Cesa-Bianchi, A. Conconi, and C.Gentile. On the generalization ability of on-line learning algorithms. *IEEE Transactions on Information Theory*, 2004.

[3] K. Crammer, O. Dekel, J. Keshet, S. Shalev-Shwartz, and Y. Singer. Online passive aggressive algorithms. *Journal of Machine Learning Research*, 2006.

[4] K. Crammer, J. Kandola, and Y. Singer. Online classification on a budget. *NIPS 16*, 2003.

[5] O. Dekel, S. Shalev-Shwartz, and Y. Singer. The Forgetron: A kernel-based perceptron on a fixed budget. *NIPS 18*, 2005.

[6] Y. Freund and R. E. Schapire. Large margin classification using the perceptron algorithm. *Machine Learning*, 37(3):277–296, 1999.

[7] S. I. Gallant. Optimal linear discriminants. *ICPR 8*, pages 849–852. IEEE, 1986.

[8] D. P. Helmbold and M. K. Warmuth. On weak learning. *Journal of Computer and System Sciences*, 50:551–573, 1995.

[9] Y. Li. Selective voting for perceptron-like on-line learning. In *ICML 17*, 2000.

[10] N. Littlestone. From on-line to batch learning. *COLT 2*, pages 269–284, July 1989.

[11] N. Littlestone and M. Warmuth. Relating data compression and learnability. Unpublished manuscript, November 1986.

[12] V. N. Vapnik. *Statistical Learning Theory*. Wiley, 1998.

[13] J. Weston, A. Bordes, and L. Bottou. Online (and offline) on a tighter budget. *AISTAT 10*, 2005.
